# A Bayesian LDA-based model for semi-supervised part-of-speech tagging

**Kristina Toutanova**
Microsoft Research
Redmond, WA
kristout@microsoft.com

**Mark Johnson**
Brown University
Providence, RI
Mark_Johnson@brown.edu

## Abstract

We present a novel Bayesian model for semi-supervised part-of-speech tagging. Our model extends the Latent Dirichlet Allocation model and incorporates the intuition that words' distributions over tags, $p(t|w)$, are sparse. In addition we introduce a model for determining the set of possible tags of a word which captures important dependencies in the ambiguity classes of words. Our model outperforms the best previously proposed model for this task on a standard dataset.

## 1 Introduction

Part-of-speech tagging is a basic problem in natural language processing and a building block for many components. Even though supervised part-of-speech taggers have reached performance of over 97% on in-domain data [1, 2], the performance on unknown in-domain words is below 90% and the performance on unknown out-of-domain words can be below 70% [3]. Additionally, few languages have a large amount of data labeled for part-of-speech. Thus it is important to develop methods that can use unlabeled data to learn part-of-speech. Research on unsupervised or partially supervised part-of-speech tagging has a long history [4, 5]. Recent work includes [6, 7, 8, 9, 10].

As in most previous work on partially supervised part-of-speech tagging, our model takes as input a (possibly incomplete) tagging dictionary, specifying, for some words, all of their possible parts of speech, as well as a corpus of unlabeled text. Our model departs from recent work on semi-supervised part-of-speech induction using sequence HMM-based models, and uses solely observed context features to predict the tags of words. We show that using this representation of context gives our model substantial advantage over standard HMM-based models.

There are two main innovations of our approach. The first is that we incorporate a sparse prior on the distribution over tags for each word, $p(t|w)$, and employ a Bayesian approach that maintains a distribution over parameters, rather than committing to a single parameter value. Previous approaches to part-of-speech tagging ([9, 10]) also use sparse priors and Bayesian inference, but do not incorporate sparse priors directly on the $p(t|w)$ distribution. Our results demonstrate that encoding this sparse prior and employing a Bayesian approach contributes significantly to performance.

The second innovation of our approach is that we explicitly model ambiguity class (the set of part-of-speech tags a word type can appear with). We show that this also results in substantial performance improvement. Our model outperforms the best-performing previously proposed model for this task [7], with an error reduction of up to 57% when the amount of supervision is small.

The task setting is more formally as follows. Assume we are given a finite set of possible part-of-speech tags (labels) $T = \{t_1, t_2, \ldots, t_{n_T}\}$. The set of part-of-speech tags for English we experiment with has the 17 tags defined by Smith & Eisner [7], and is a coarse-grained version of the 45-tag set in the English Penn Treebank. We are also given a dictionary which specifies the ambiguity classes $s \subseteq T$ for a subset of the word types $w$. The ambiguity class of a word type is the set of all of its

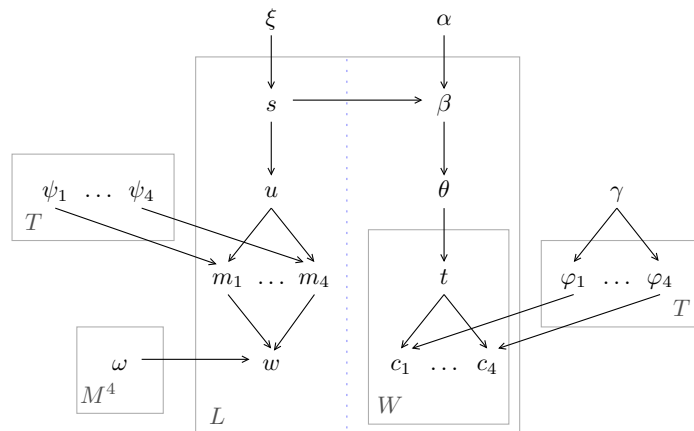

$$
\begin{array}{llll}
s_i & \mid \xi & \sim \text{MULTI}(\xi), & i = 1, \ldots, L \\
u_i & \mid s_i & \sim \text{UNIFORM}(s_i) & i = 1, \ldots, L \\
m_{j,i} & \mid u_i, \psi_j & \sim \text{MULTI}(\psi_{j,u_i}), & i = 1, \ldots, L, j = 1, \ldots, 4 \\
w_i & \mid m_i, \omega & \sim \text{MULTI}(\omega_{m_i}), & i = 1, \ldots, L \\
\beta_i & \mid \alpha, s_i & = \text{SUBSET}(\alpha, s_i), & i = 1, \ldots, L \\
\theta_i & \mid \beta_i & \sim \text{DIR}(\beta_i), & i = 1, \ldots, L \\
t_{i,j} & \mid \theta_i & \sim \text{MULTI}(\theta_i), & i = 1, \ldots, L, j = 1, \ldots, W_i \\
\varphi_{k,\ell} & \mid \gamma & \sim \text{DIR}(\gamma), & k = 1, \ldots, 4, \ell = 1, \ldots, T \\
c_{k,i,j} & \mid t_{i,j}, \varphi_k & \sim \text{MULTI}(\varphi_{k,t_{i,j}}), & i = 1, \ldots, L, j = 1, \ldots, W_i, k = 1, \ldots, 4
\end{array}
$$

Figure 1: A graphical model for the tagging model. In this model, each word type $w$ is associated with a set $s$ of possible parts-of-speech (ambiguity class), and each of its tokens is associated with a part-of-speech tag $t$, which generates the context words $c$ surrounding that token. The ambiguity class $s$ also generates the morphological features $m$ of the word type $w$ via a hidden tag $u \in s$. The dotted line divides the model into the ambiguity class model (on the left) and the word context model (on the right).

possible tags. For example, the dictionary might specify that *walks* has the ambiguity class $\{N, V\}$ which means that *walks* can never have a tag which is not an *N* or a *V*. Additionally, we are given a large amount of unlabeled natural language text. The task is to label each word token with its correct part-of-speech tag in the corresponding context.

This task formulation corresponds to a problem in computational linguistics that frequently arises in practice, because the only available resources for many languages consist of a manually constructed dictionary and a text corpus. Note that it differs from the standard semi-supervised learning setting, where we are given a small amount of labeled data and a large amount of unlabeled data. In the setting we study, we are never given labeled data, but are given instead constraints on possible tags of some words (in the form of a dictionary).[1]

## 2 Graphical model

Our model is shown in Figure 1. In the figure, $T$ is the set of part-of-speech tags, $L$ is the set of word types (i.e., the set of different orthographic forms), $W$ is the set of tokens (i.e., occurrences) of the word type $w$, and $M^4$ is the set of four-element morphological feature vectors described below.

This is a generative model for a sequence of word tokens in a text corpus along with part-of-speech tags for all tokens, ambiguity classes for word types and other hidden variables. To generate the text corpus, the model generates the instances of every word type together with their contexts in

turn. The generation of a word type and all of its occurrences can be decomposed into two steps, corresponding to the left and right parts of the model: the ambiguity class model, and the word context model (separated by a dotted line in the figure).

For every word type $w_i \in L$ (plate $L$ in the figure), in the first step the model generates an ambiguity class $s_i \subseteq T$ of possible parts of speech. The ambiguity class $s_i$ is the set of parts-of-speech that tokens of $w_i$ can be labeled with. Our dictionary specifies $s_i$ for some but not all word types $w_i$. The ambiguity class $s_i$ is generated by a multinomial over $2^T$ with parameters $\xi$, with support on the different values for $s$ observed in the dictionary. The ambiguity class $s_i$ for $w_i$ generates four different morphological features $m_{1,i}, \ldots, m_{4,i}$ of $w_i$ representing the suffixes, capitalization, etc., of the orthographic form of $w_i$. These are generated by multinomials with parameters $\psi_{1,u}, \ldots, \psi_{4,u}$ respectively, where $u \in s$ is a hidden variable generated by a uniform distribution over the members of $s$. For completeness we generate the full surface form of the word type $w_i$ from a multinomial distribution selected by its morphology features $m_{1,i}, \ldots, m_{4,i}$. But since the morphology features are always observed (they are determined by $w_i$'s orthographic form), we ignore this part of the model. We discuss the ambiguity class model in detail in Section 3.1.

In the second step the word context model generates all instances $w_{i,j}$ of $w_i$ together with their part-of-speech tags $t_{i,j}$ and context words (plate $W$ in the figure). This is done by first choosing a multinomial distribution $\theta_i$ over the tags in the set $s_i$, which is drawn from a Dirichlet with parameters $\beta_i$ and support $s_i$, where $\beta_{i,t} = \alpha_t$ for $t \in s$. That is, $s_i$ identifies the subset of $T$ to receive support in $\beta_i$, but the value of $\beta_{i,t}$ for $t \in s_i$ is specified by $\alpha_t$. Given these variables, all tokens $w_{i,j}$ of the word $w_i$ together with their contexts are generated by first choosing a part-of-speech tag $t_{i,j}$ from $\theta_i$ and then choosing context words $c_{k,i,j}$ preceding and following the word token $w_{i,j}$ according to tag-specific (depending on $t_{i,j}$) multinomial distributions. The context of a word token $c_{1,i,j} \ldots, c_{4,i,j}$ consists of the two preceding and two following words. For example, for the sentence *He often walks to school*, the context words of that instance of *walks* are $c_1$=*He*, $c_2$=*often*, $c_3$=*to*, and $c_4$=*school*. This representation of the context has been used previously by unsupervised models for part-of-speech tagging in different ways [4, 8]. Each context word $c_{k,i,j}$ is generated by a multinomial with parameters $\varphi_{k,t_{i,j}}$, where each $\varphi_{k,t}$ is in turn generated by a Dirichlet with parameters $\gamma$. The parameters $\varphi_{k,t}$ are generated once for the whole corpus as indicated in the figure.

A sparse Dirichlet prior on $\theta_i$ with parameter $\alpha < 1$ allows us to exploit the fact that most words have a very frequent predominant tag, and their distribution over tags $p(t|w)$ is sparse. To verify this, we examined the distribution of the 17-label tag set in the WSJ Penn Treebank. A classifier that always chooses the most frequent tag for every word type, without looking at context, is 90.9% accurate on ambiguous words, indicating that the distribution is heavily skewed.

Our model builds upon the Latent Dirichlet Allocation (LDA) model [11] by extending it in several ways. If we assume that the only possible ambiguity class $s$ for all words is the set of all tags (and thus remove the ambiguity class model because it becomes irrelevant), and if we simplify our word context model to generate only one context word (say the word in position $-1$), we would end up with the LDA model. In this simplified model, we could say that for every word type $w_i$ we have a document consisting of all word tokens that occur in position $-1$ of the word type $w_i$ in the corpus. Each context word $c_{i,j}$ in $w_i$'s document is generated by first choosing a tag (topic) from a word (document) specific distribution $\theta_i$ and then generating the word $c_{i,j}$ from a tag (topic) specific multinomial. The LDA model incorporates the same kind of Dirichlet priors on $\theta$ and $\varphi$ that our model uses. The additional power of our model stems from the model of ambiguity classes $s_i$ which can take advantage of the information provided by the dictionary, and from the incorporation of multiple context features.

Finally, we note that our model is deficient, because the same word token in the corpus is independently generated multiple times (e.g., each token will appear in the context of four other words and will be generated four times). Even though this is a theoretical drawback of the model, it remains to be seen whether correcting for this deficiency (e.g., by re-normalization) would improve tagging performance. Models with similar deficiencies have been successful in other applications (e.g. the model described in [12], which achieved substantial improvements over the previous state-of-the-art in unsupervised parsing).

# 3 Parameter estimation and tag prediction

Here we discuss our method of estimating the parameters of our model and making predictions, given an (incomplete) tagging dictionary and a set of natural language sentences.

We train the parameters of the ambiguity class model, $\xi$, $\psi$, and $\omega$, separately from the parameters of the word context model: $\alpha, \theta, \gamma$, and $\varphi$. This is because the two parts of the model are connected only via the variables $s_i$ (the ambiguity classes of words), and when these ambiguity classes are given the two sets of parameters are completely decoupled. The dictionary gives us labeled training examples for the ambiguity class model, and we train the parameters of the ambiguity class model only from this data (i.e., the word types in the dictionary). After training the ambiguity class model from the dictionary we fix its parameters and estimate the word context model given these parameters.

## 3.1 Ambiguity class model: details and parameter estimation

Our ambiguity class model captures the strong regularities governing the possible tags of a word type. Empirically we observe that the number of occurring ambiguity classes is very small relative to the number of possible ambiguity classes. For example, in the WSJ Penn Treebank data, the $49,206$ word types belong to $118$ ambiguity classes. Modeling these (rather than POS tags directly) constrains the model to avoid assignments of tags to word tokens which would result in improbable ambiguity classes for word types. A related intuition has been used in other contexts before, e.g. [13, 14], but without directly modeling ambiguity classes. The ambiguity class model contributes to biasing $p(t|w)$ toward sparse distributions as well, because most ambiguity classes have very few elements. For example, the top ten most frequent ambiguity classes in the complete dictionary consist of one or two elements.

The ambiguity class of a word type can be predicted from its surface morphological features. For example the suffix *-s* of *walks* indicates that an ambiguity class of $\{N, V\}$ is likely for this word. The four morphological features which we used for the ambiguity class model were: a binary feature indicating whether the word is capitalized, a binary feature indicating whether the word contains a hyphen, a binary feature indicating whether the word contains a digit character, and a nominal feature indicating the suffix of a word. We define the suffix of a word to be the longest character suffix (up to three letters) which occurs as a suffix of sufficiently many word types.[2]

We train the ambiguity class model on the set of word types present in the dictionary. We set the multinomial parameters $\psi_{k,l}$ and $\xi$ to maximize the joint likelihood of these word types and their morphological features. Maximum likelihood estimation for $\psi$ is complicated by the hidden variable $u_i$ which selects a tag form the ambiguity class with uniform distribution.

$$P(s, m_1, m_2, m_3, m_4 | \psi, \xi) = P(s|\xi) \sum_{u \in s} P(u|s) \prod_{j=1}^{4} P(m_j | \psi_{j,l}).$$

We fix the probability $P(u|s) = 1/|s|$ to the uniform distribution over tags in $s$. We estimate the $\xi$ parameters using maximum likelihood estimation with add-1 (Laplace) smoothing and we train the $\psi$ parameters using EM (with add-1 smoothing in the M-step).

## 3.2 Parameter estimation for the word context model and prediction given complete dictionary

We restrict our attention at first to the setting where a complete tagging dictionary is given. The incomplete dictionary generalization is discussed in Section 3.3. When every word is in the dictionary, the ambiguity class $s_i$ for each word type $w_i$ is specified by the tagging dictionary, and the ambiguity class model becomes irrelevant. The relevant parameters of the model in this setting are $\alpha, \theta, \gamma$, and $\varphi$. The contexts of word instances $c_{k,i,j}$ and the ambiguity classes $s_i$ are observed.

We integrate over all hidden variables except the uniform Dirichlet parameters $\alpha$ and $\gamma$. We set $\gamma = 1$ and we use Empirical Bayes to estimate $\alpha$ by maximizing the likelihood of the observed data given $\alpha$ and the ambiguity classes $s_i$. Note that if the ambiguity classes $s_i$ and $\alpha$ are given, $\beta_i$ is fixed. Below we use **c** to denote the vector of all contexts of all word instances, and **s** the vector of ambiguity classes for all word types. We use $\varphi$ to denote the vector of all multinomials $\varphi_{k,l}$, $\theta$ to

denote the vector of all $\theta_i$ and $t$ to denote the vector of all tag sequences $t_i$ for word types $w_i$. The likelihood we would like to maximize is:

$$L(\mathbf{c}|\mathbf{s},\alpha,\gamma) = \int P(\varphi|\gamma) \prod_{i=1}^{L} \int P(\theta_i|\beta_i) \prod_{j=1}^{W_i} \sum_{l=1}^{T} \left(\theta_{i,l} \prod_{k=1}^{4} P(c_{k,i,j}|\varphi_{k,l})\right) d\theta_i d\varphi$$

$$P(\varphi|\gamma) = \prod_{k=1}^{4} \prod_{l=1}^{T} \text{DIR}(\varphi_{k,l}|\gamma)$$

Since exact inference is intractable, we use a variational approximation to the posterior distribution of the hidden variables given the data and maximize instead of the exact log-likelihood, a lower bound given by the variational approximation. This variational approximation is also used for finding the most likely assignment of the part-of-speech tag variables to instances of words.

More specifically, the variational approximation has analogous form to the approximation used for the LDA model [11]. It depends on variational parameters $\lambda_{k,l}$, $\eta_i$, and $\upsilon_{i,j}$.

$$Q(\varphi,\theta,t|\lambda,\eta,\upsilon) = \prod_{k=1}^{4} \prod_{l=1}^{T} \text{DIR}(\varphi_{k,l}|\lambda_{k,l}) \prod_{i=1}^{L} \text{DIR}(\theta_i|\eta_i) \prod_{j=1}^{W_i} P(t_{i,j}|\upsilon_{i,j})$$

This distribution is an approximation to the posterior distribution of the hidden variables: $P(\varphi,\theta,t|\mathbf{c},\mathbf{s},\alpha,\gamma)$. As we can see, according to the $Q$ distribution, the variables $\varphi$, $\theta$, and $t$ are independent. Each $\varphi_{k,l}$ is distributed according to a Dirichlet distribution with variational parameters $\lambda_{k,l}$, each $\theta_i$ is also Dirichlet with parameters $\eta_i$ and each tag $t_{i,j}$ is distributed according to a multinomial $\upsilon_{i,j}$. We obtain the variational parameters by maximizing the following lower bound on the log-likelihood of the data (the dependence of $Q$ on the variational parameters is not shown below for simplicity): $E_Q\left[\log P(\varphi,\theta,t,\mathbf{c}|\mathbf{s},\alpha,\gamma)\right] - E_Q\left[\log Q(\varphi,\theta,t)\right]$

We use an iterative maximization algorithm for finding the values of the variational parameters. We do not describe it here due to space limitations, but it is analogous to the one used in [11]. Given fixed variational parameters $\lambda_{k,l}$ we maximize with respect to the variational parameters $\eta_i$ and $\upsilon_{i,j}$ corresponding to word types and their instances. Then keeping the latter parameters fixed, we maximize with respect to $\lambda_{k,l}$. We repeat until the change in the variational bound falls below a threshold. On our dataset, about 100 iterations of the outer loop for maximizing with respect to $\lambda_{k,l}$ were necessary. Given a variational distribution $Q$ we can maximize the lower bound on the log-likelihood with respect to $\alpha$. Since $\alpha$ is determined by a single real-valued parameter, we maximized with respect to $\alpha$ using a simple grid search.

For predicting the tags $t_{i,j}$ of word tokens we use the same approximate posterior distribution $Q$. Since according to $Q$ all tags $t_{i,j}$ are independent given the variational parameters: $Q(t_i|\upsilon_i) = \prod_{j=1}^{W_i}(t_{i,j}|\upsilon_{i,j})$, finding the most likely assignment is straightforward.

### 3.3 Parameter estimation for the word context model and prediction with incomplete dictionary

So far we have described the training of the parameters of the word context model in the setting where for all words, the ambiguity classes $s_i$ are known and these variables are observed. When the ambiguity classes $s_i$ are unknown for some words in the dataset, they become additional hidden variables, and the hidden variables in the word context model become dependent on the morphological features $m_i$ and the parameters of the ambiguity class model. Denote the vector of ambiguity classes for the known (in the dictionary) word types by $\mathbf{s_d}$ and the ambiguity classes for the unknown word types by $\mathbf{s_u}$. The posterior distribution over the hidden variables of interest given the observations becomes: $P(\varphi,\theta,t,\mathbf{s_u}|\mathbf{s_d},\mathbf{m_u},\mathbf{c},\alpha,\gamma)$, where $\mathbf{m_u}$ are the morphological features of the unknown word types.

To perform inference in this setting we extend the variational approximation to account for the additional hidden variables. Before we had, for every word type, a variational distribution over the hidden variables corresponding to that word type:

$$Q(\theta_i,t_i|\eta_i,\upsilon_{i,j}) = \text{DIR}(\theta_i|\eta_i) \prod_{j=1}^{W_i} P(t_{i,j}|\upsilon_{i,j})$$

We now introduce a variational distribution including new hidden variables $s_i$ for unknown words.

$$Q(\theta_i,t_i,s_i|m_i,\eta_{i,s},\upsilon_{i,j,s}) = P(s_i|m_i)\text{DIR}(\theta_i|\eta_{i,s}) \prod_{j=1}^{W_i} P(t_{i,j}|\upsilon_{i,j,s_i})$$

That is, for each possible ambiguity class $s_i$ of an unknown word $w_i$ we introduce variational parameters specific to that ambiguity class. Instead of single variational parameters $\theta_i$ and $\upsilon_{i,j}$ for a word with known $s_i$, we now have variational parameters $\{\theta_{i,s}\}$ and $\{\upsilon_{i,j,s}\}$ for all possible values $s$ of $s_i$. For simplicity, we use the probability $P(s_i|m_i) = P(s_i|m_i, \xi, \psi)$ from the morphology-based ambiguity class model in the approximating distribution rather than introducing new variational parameters and learning this distribution.[3] We adapt the algorithm to estimate the variational parameters. The derivation is slightly complicated by the fact that $s_i$ and $\theta_i$ are not independent according to $Q$ (this makes sense because $s_i$ determines the dimensionality of $\theta_i$), but the derived iterative algorithm is essentially the same as for our basic model, if we imagine that an unknown word type $w_i$ occurs with each of its possible ambiguity classes $s_i$ a fractional $p(s_i|m_i)$ number of times.

For predicting tag assignments for words according to this extended model, we use the same algorithm as described in Section 3.2, for word types whose ambiguity classes $s_i$ are known. For words with unknown ambiguity classes, we need to maximize over ambiguity classes as well as tag assignments. We use the following algorithm to obtain a slightly better approximation than the one given by the variational distribution $Q$. For each possible tag set $s_i$, we find the most likely assignment of tags given that ambiguity class $t^*(s_i)$, using the variational distribution as in the case of known ambiguity classes. We then choose an ambiguity class and an assignment of tags according to:

$$s^* = \arg\max_{s_i} P(s_i|m_i, \psi, \xi) P(t^*(s_i), c_i|s_i, \mathcal{D}, \alpha, \gamma) \text{ and } t = t^*(s^*).$$

We compute $P(t^*(s_i), c_i|s_i, \mathcal{D}, \alpha, \gamma)$ by integrating with respect to the word context distributions $\varphi$ whose approximate posterior given the data is Dirichlet with parameters $\lambda_{k,l}$, and by integrating with respect to $\theta_i$ which are Dirichlet with parameters $\alpha$ and dimensionality given by $s_i$.

## 4  Experimental Evaluation

We evaluate the performance of our model in comparison with other related models. We train and evaluate the model in three different settings. In the first setting, a complete tagging dictionary is available, and in the other two settings the coverage of the dictionary is greatly reduced.

The tagging dictionary was constructed by collecting for each word type, the set of parts-of-speech with which it occurs in the annotated WSJ Penn Treebank, including the test set. This method of constructing a tag dictionary is arguably unrealistic but has been used in previous research [7, 9, 6] and provides a reproducible framework for comparing different models. In the complete dictionary setting, we use the ambiguity class information for all words, and in the second and third setting we remove from the dictionary all word types that have occurred with frequency less than 2 and less than 3, respectively, in the test set of 1,005 sentences. The complete tagging dictionary contains entries for $49,206$ words. The dictionary obtained with cutoff of 2 contains 2,141 words, and the one with cutoff of 3 contains 1,249 words. We train the model on the whole (unlabeled) WSJ Penn Treebank, consisting of 49,208 sentences. We evaluate performance on a set of 1,005 sentences, which is a subset of the training data and is the same test set used by [7, 9].

To see how much removing information from the dictionary impacts the hardness of the problem we can look at the accuracy of a classifier choosing a tag at random from the possible tags of words, shown in the column **Random** of Table 1. Results for the three settings are shown in the three rows of Table 1. In addition to the **Random** baseline, we include the results of a frequency baseline, **Freq**, in which for each word, we choose the most frequent tag from its set of possible tags.[4] This baseline uses the same amount of partial supervision as our models. If labeled corpus data were available, a model which assigns the most frequent tag to each word by using $\hat{p}(t|w)$ would do much better.

The models in the table are:

**LDA** is the model proposed in this paper, excluding the ambiguity class model. The ambiguity class model is irrelevant when a compete dictionary is available because all $s_i$ are observed. In the other two settings for the LDA model we assume that $s_i$ is the complete ambiguity class (all 17 tags)

| Dictionary coverage | LDA | LDA + AC | PLSA | PLSA +AC | CE (S&E) +spelling | Bayesian HMM (G&G) | ML HMM (G&G) | Random | Freq |
|---|---|---|---|---|---|---|---|---|---|
| complete | 93.4 | **93.4** | 89.7 | 89.7 | 88.7 (91.9) | 87.3 | 83.2 | 69.5 | 64.8 |
| count $\geq 2$ | 87.4 | **91.2** | 83.4 | 87.8 | 79.5 (90.3) | 79.6 | 70.6 | 56.6 | 64.8 |
| count $\geq 3$ | 85.0 | **89.7** | 80.2 | 85.9 | 78.4 (89.5) | 71.0 | 65.5 | 51.0 | 62.9 |

Table 1: Results from minimally supervised POS-tagging models.

for words which are not in the dictionary and do not attempt to predict a more specific ambiguity class. The estimated parameter $\alpha$ for the tag prior was $0.5$ for the complete dictionary setting, and $0.2$ for the other two settings, encouraging sparse distributions. For this model we estimate the variational parameters $\lambda_{k,l}$ and the Dirichlet parameter $\alpha$ to maximize the variational bound on the log-likelihood of the word types which are in the dictionary only. We found that including unknown word types was detrimental to performance.

**LDA+AC** is our full model including the model of ambiguity classes of words given their morphological features. As mentioned above, this augmented model differs from **LDA** only when the dictionary is incomplete. We trained this model on all word types as discussed in Section 3.3. The estimated $\alpha$ parameters for this model in the three dictionary settings were $0.5$, $0.1$, and $0.1$, respectively.

**PLSA** is the model analogous to LDA, which has the same structure as our word context model, but excludes the Bayesian components. We include this model in the comparison in order to evaluate the effect on performance of the sparse prior and the integration over model parameters. This model is similar to the PLSA model for text documents [15]. The PLSA model does not have a prior on the word-specific distributions over tags $\theta_i = p(t|w_i)$ and it does not have a prior distribution on the topic-specific multinomials for context words $\varphi_{k,l}$. For this model we find maximum likelihood estimates for these parameters by applying an EM algorithm. We do add-1 smoothing for $\varphi_{k,l}$ in the M step, because even though this is not theoretically justified for this mixture model, it is frequently used in practice and helps prevent probabilities of zero for possible events. PLSA does not include the ambiguity class model for $s_i$ and as in the LDA model, word types not in the dictionary were assumed to have ambiguity classes containing all 17 tags. **PLSA+AC** extends the PLSA model by the inclusion of the ambiguity class model.

**CE+spelling** (S&E) is the sequence model for semi-supervised part-of-speech tagging proposed in [7], based on an HMM-structured model estimated using contrastive estimation. This is the state-of-the-art model for semi-supervised tagging using an incomplete dictionary. In the table we show actual performance and oracle performance for this model (oracle performance is in brackets).The oracle is obtained by testing models with different values of a smoothing hyper-parameter on the test set and choosing the model with the best accuracy. Even though there is only one real-valued hyper-parameter, the accuracies of models using different values can vary by nearly ten accuracy points and it is thus more fair to compare our results to the non-oracle result, until a better criterion for setting the hyper-parameters using only the partial supervision is found. The results shown in the table are for a model which incorporates morphological features.

**Bayesian HMM** (G&G) is a fully Bayesian HMM model for semi-supervised part-of-speech tagging proposed in [9], which incorporates sparse Dirichlet priors on $p(w|t)$ of word tokens given part of speech tags and $p(t_i|t_{i-1}, t_{i-2})$ of transition probabilities in the HMM. We include this model in the comparison, because it uses sparse priors and Bayesian inference as our LDA model, but using a different structure of the model. [9] showed that this model outperforms significantly a non-Bayesian HMM model, whose results we show as well.

**ML HMM** (G&G) is the maximum likelihood version of a trigram HMM for semi-supervised part-of-speech tagging. Results for this model have been reported by other researchers as well [7, 6]. We use the performance numbers reported in [9] because they have used the same data sets for testing.

The last two models do not use spelling (morphological) features. We should note that even though the same amount of supervision in the form of a tagging dictionary is used by all compared models, the HMM and CE models whose results are shown in the Table have been trained on less unsupervised natural language text: they have been trained using only the test set of 1,005 sentences. However, there is no reason one should limit the amount of unlabeled data used and in addition,

other results reported in [7] and [9] show that accuracy does not seem to improve when more unlabeled data are used with these models.

There are several points to note about the experimental results. First, the fact that PLSA substantially outperforms ML HMM (and even the Bayesian HMM) models shows that predicting the tags of words from a window of neighboring word tokens and modeling the $P(t|w)$ distribution directly results in an advantage over HMMs with maximum likelihood or Bayesian estimation. This is consistent with the success of other models that used word context for part-of-speech prediction in different ways [4, 8]. Second, the Bayesian and sparse-prior components of our model do indeed contribute substantially to performance, as illustrated by the performance of LDA compared to that of PLSA. LDA achieves an error reduction of up to 36% over PLSA. Third, our ambiguity class model results in a significant improvement as well; LDA+AC reduces the error of LDA by up to 31%. PLSA+AC similarly reduces the error of PLSA. Finally, our complete model outperforms the state-of-the-art model CE+spelling. It reduces the error of the non-oracle models by up to 57% and also outperforms the oracle models.

We compared the performance of our model to that of state-of-the-art models applied in the same setting. It will also be interesting to compare our model to the one proposed in [8], which was applied in a different partial supervision setting. In their setting a small set of example word types (which they call prototypes) are provided for each possible tag (only three prototypes per tag were specified). Their model achieved an accuracy of 82.2% on a similar dataset. We can not directly compare the performance of our model to theirs, because our model would need prototypes for every ambiguity class rather than for every tag. In future work we will explore whether a very small set of prototypical ambiguity classes and corresponding word types can achieve the performance we obtained with an incomplete tagging dictionary. Another interesting direction for future work is applying our model to other NLP disambiguation tasks, such as named entity recognition and induction of deeper syntactic or semantic structure, which could benefit from both our ambiguity class model and our word context model.

## Footnotes

[1]For some words, the dictionary specifies only one possible tag, e.g. *information* $\rightarrow \{N\}$, in which case all instances of *information* can be assumed labeled with the tag *N*. However these constraints are not sufficient to result in fully labeled sentences.

[2]A suffix occurs with sufficiently many word types if its type-frequency rank is below 100.

[3]We also limit the number of possible ambiguity classes per word to the three most likely ones and renormalize the probability mass among them.

[4]Frequency of tags is unigram frequency of tags $\hat{p}(t)$ by token in the unlabeled data. Since the tokens in the corpus are not actually labeled we compute the frequency by giving fractional counts to each possible tag of words in the dictionary. Only the words present in the dictionary were used for computing $\hat{p}(t)$.

## References

[1] Kristina Toutanova, Dan Klein, and Christopher D. Manning. Feature-rich part-of-speech tagging with a cyclic dependency network. In *Proceedings of HLT-NAACL 03*, 2003.

[2] Michael Collins. Discriminative training methods for hidden markov models: Theory and experiments with perceptron algorithms. In *EMNLP*, 2002.

[3] John Blitzer, Ryan McDonald, and Fernando Pereira. Domain adaptation with structural correspondence learning. In *EMNLP*, 2006.

[4] Hinrich Schütze. Distributional part-of-speech tagging. In *EACL*, 1995.

[5] Bernard Merialdo. Tagging english text with a probabilistic model. In *ICASSP*, 1991.

[6] Michele Banko and Robert C. Moore. Part of Speech tagging in context. In *COLING*, 2004.

[7] Noah A. Smith and Jason Eisner. Contrastive estimation: Training log-linear models on unlabeled data. In *ACL*, 2005.

[8] Aria Haghighi and Dan Klein. Prototype-driven learning for sequence models. In *HLT-NAACL*, 2006.

[9] Sharon Goldwater and Thomas L. Griffiths. A fully Bayesian approach to unsupervised Part-of-Speech tagging. In *ACL*, 2007.

[10] Mark Johnson. Why doesn't EM find good HMM POS-taggers. In *EMNLP*, 2007.

[11] David Blei, Andrew Ng, and Michael Jordan. Latent dirichlet allocation. *Journal of Machine Learning Research*, 3:993–1022, 2003.

[12] Dan Klein and Christopher D. Manning. Natural language grammar induction using a constituent-context model. In *NIPS 14*, 2002.

[13] Jenny Rose Finkel, Trond Grenager, and Christopher Manning. Incorporating non-local information into information extraction systems by Gibbs sampling. In *ACL*, 2005.

[14] Tetsuji Nakagawa and Yuji Matsumoto. Guessing parts-of-speech of unknown words using global information. In *ACL*, 2006.

[15] Thomas Hofmann. Probabilistic latent semantic analysis. In *UAI*, 1999.

